# Analyzing Cross Connected Networks

**Thomas R. Shultz**
Department of Psychology &
McGill Cognitive Science Centre
McGill University
Montréal, Québec, Canada H3A 1B1
shultz@psych.mcgill.ca

and

**Jeffrey L. Elman**
Center for Research on Language
Department of Cognitive Science
University of California at San Diego
LaJolla, CA 92093-0126 U.S.A.
elman@crl.ucsd.edu

## Abstract

The non-linear complexities of neural networks make network solutions difficult to understand. Sanger's *contribution analysis* is here extended to the analysis of networks automatically generated by the cascade-correlation learning algorithm. Because such networks have cross connections that supersede hidden layers, standard analyses of hidden unit activation patterns are insufficient. A *contribution* is defined as the product of an output weight and the associated activation on the sending unit, whether that sending unit is an input or a hidden unit, multiplied by the sign of the output target for the current input pattern. Intercorrelations among contributions, as gleaned from the matrix of contributions x input patterns, can be subjected to principal components analysis (PCA) to extract the main features of variation in the contributions. Such an analysis is applied to three problems, continuous XOR, arithmetic comparison, and distinguishing between two interlocking spirals. In all three cases, this technique yields useful insights into network solutions that are consistent across several networks.

## 1   INTRODUCTION

Although neural network researchers are typically impressed with the performance achieved by their learning networks, it often remains a challenge to explain or even characterize such performance. The latter difficulties stem principally from the complex non-linear properties of neural nets and from the fact that information is encoded in a form that is distributed across many weights and units. The problem is exacerbated by the fact that multiple nets generate unique solutions depending on variation in both starting states and training patterns.

Two techniques for network analysis have been applied with some degree of success, focusing respectively on either a network's weights or its hidden unit activations. Hinton (e.g., Hinton & Sejnowski, 1986) pioneered a diagrammatic analysis that involves plotting a network's learned weights. Occasionally, such diagrams yield interesting insights but often, because of the highly distributed nature of network representations, the most notable features of such analyses are the complexity of the pattern of weights and its variability across multiple networks learning the same problem.

Statistical analysis of the activation patterns on the hidden units of three layered feed-forward nets has also proven somewhat effective in understanding network performance. The relations among hidden unit activations, computed from a matrix of hidden units x input patterns, can be subjected to either cluster analysis (Elman, 1990) or PCA (Elman, 1989) to determine the way in which the hidden layer represents the various inputs. However, it is not clear how this technique should be extended to multi-layer networks or to networks with cross connections.

Cross connections are direct connections that bypass intervening hidden layers. Cross connections typically speed up learning when used in static back-propagation networks (Lang & Witbrock, 1988) and are an obligatory and ubiquitous feature of some generative learning algorithms, such as cascade-correlation (Fahlman & Lebiere, 1990). Generative algorithms construct their own network topologies as they learn. In cascade-correlation, this is accomplished by recruiting new hidden units into the network, as needed, installing each on a separate layer. In addition to layer-to-layer connections, each unit in a cascade-correlation network is fully cross connected to all non-adjacent layers downstream. Because such cross connections carry so much of the work load, any analysis restricted to hidden unit activations provides a partial picture of the network solution at best.

Generative networks seem to provide a number of advantages over static networks, including more principled network design, leaner networks, faster learning, and more realistic simulations of human cognitive development (Fahlman & Lebiere, 1990; Shultz, Schmidt, Buckingham, & Mareschal, in press). Thus, it is important to understand how these networks function, even if they seem impervious to standard analytical tools.

## 2    CONTRIBUTION ANALYSIS

One analytical technique that might be adapted for multi-layer, cross connected nets is *contribution analysis* (Sanger, 1989). Sanger defined a *contribution* as the triple product of an output weight, the activation of a sending unit, and the sign of the output target for that input. He argued that contributions are potentially more informative than either weights alone or hidden unit activations alone. A large weight may not contribute much if it is connected to a sending unit with a small activation. Likewise, a large sending activation may not contribute much if it is connected via a small weight. In contrast, considering a full contribution, using both weight and sending activation, would more likely yield valid comparisons.

Sanger (1989) applied contribution analysis to a small version of NETtalk, a net that learns to convert written English into spoken English (Sejnowski & Rosenberg, 1987). Sanger's analysis began with the construction of an output unit x hidden unit x input pattern array of contributions. Various two-dimensional slices were taken from this three-dimensional array, each representing a particular output unit or a particular hidden unit. Each two-dimensional slice was then subjected to PCA, yielding information about either distributed or local hidden unit responsibilities, depending on whether the focus was on an individual output unit or individual hidden unit, respectively.

## 3    CONTRIBUTION ANALYSIS FOR MULTI-LAYER, CROSS CONNECTED NETS

We adapted contribution analysis for use with multi-layered, cross connected cascade-correlation nets. Assume a cascade-correlation network with $j$ units (input units + hidden units) and $k$ output units, being trained with $i$ input patterns. There are $j$ x $k$ output weights in such a network, where an output weight is defined as any weight connected to

an output unit. A contribution $c$ for a particular $ijk$ combination is defined as

$$c_{ijk} = w_{jk} \, a_{ij} \, 2t_{ki} \qquad\qquad (1)$$

where $w_{jk}$ is the weight connecting sending unit $j$ with output unit $k$, $a_{ij}$ is the activation of sending unit $j$ given input pattern $i$, and $t_{ki}$ is the target for output unit $k$ given input pattern $i$. The term $2t_{ki}$ adjusts the sign of the contribution so that it provides a measure of correctness. That is, positive contributions push the output activation towards the target, whereas negative contributions push the output activation away from the target. In cascade-correlation, sigmoid output units have targets of either -0.5 or +0.5. Hence, multiplying a target by 2 yields a positive sign for positive targets and a negative sign for negative targets. Our term $2t_{ki}$ is analogous to Sanger's (1989) term $2t_{ik} - 1$, which is appropriate for targets of 0 and 1, commonly used in back-propagation learning.

In contrast to Sanger's (1989) three-dimensional array of contributions (output unit x hidden unit x input pattern), we begin with a two-dimensional output weight $(k * j)$ x input pattern $(i)$ array of contributions. This is because we want to include all of the contributions coming into the output units, including the cross connections from more than one layer away. Since we begin with a two-dimensional array, we do not need to employ the somewhat cumbersome slicing technique used by Sanger to isolate particular output or hidden units. Nonetheless, as will be seen, our technique does allow the identification of the roles of specific contributions.

## 4    PRINCIPAL COMPONENTS ANALYSIS

Correlations among the various contributions across input patterns are subjected to PCA. PCA is a statistical technique that identifies significant dimensions of variation in a multi-dimensional space (Flury, 1988). A component is a line of closest fit to a set of points in multi-dimensional space. The goal of PCA is to summarize a multivariate data set using as few components as possible. It does this by taking advantage of possible correlations among the variables (contributions, in our case).

We apply PCA to contributions, as defined in Equation 1, taken from networks learning three different problems: continuous XOR, arithmetic comparisons, and distinguishing between interlocking spirals. The contribution matrix for each net, as described in section 3, is subjected to PCA using 1.0 as the minimum eigenvalue for retention. Varimax rotation is applied to improve the interpretability of the solution. Then the *scree* test is applied to eliminate components that fail to account for much of the variance (Cattell, 1966). In cases where components are eliminated, the analysis is repeated with the correct number of components, again with a varimax rotation. Component scores for the retained components are plotted to provide an indication of the function of the components. Finally, component loadings for the various contributions are examined to determine the roles of the contributions from hidden units that had been recruited into the networks.

## 5    APPLICATION TO THE CONTINUOUS XOR PROBLEM

The simplicity of binary XOR and the small number of training patterns (four) renders application of contribution analysis superfluous. However, it is possible to construct a continuous version of the XOR problem that is more suitable for contribution analysis. We do this by dividing the input space into four quadrants. Input values are incremented in steps of 0.1 starting from 0.0 up to 1.0, yielding 100 $x$, $y$ input pairs. Values of $x$ up to 0.5 combined with values of $y$ above 0.5 produce a positive output target (0.5), as do values of $x$ above 0.5 combined with values of $y$ below 0.5. Input pairs in the other two quadrants yield a negative output target (-0.5).

Three cascade-correlation nets are trained on this problem. Each of the three nets generates a unique solution to the continuous XOR problem, with some variation in number of hidden units recruited. PCA of contributions yields different component loadings across the three nets and different descriptions of components. Yet with all of that variation in detail, it is apparent that all three nets make the same three distinctions that are afforded by the training patterns. The largest distinction is that which the nets are explicitly trained to make, between positive and negative outputs. Two components are sufficient to describe the representations. Plots of rotated component scores for the 100 training patterns cluster into four groups of 25 points, each cluster corresponding to one of the four quadrants described earlier. Component loadings for the various contributions on the two components indicate that the hidden units play an interactive and distributed role in separating the input patterns into their respective quadrants.

## 6    APPLICATION TO COMPARATIVE ARITHMETIC

A less well understood problem than XOR in neural net research is that of arithmetic operations, such as addition and multiplication. What has a net learned when it learns to add, or to multiply, or to do both operations? The non-linear nature of multiplication makes it particularly interesting as a network analysis problem. The fact that several psychological simulations using neural nets involve problems of linear and non-linear arithmetic operations enhances interest in this sort of problem (McClelland, 1989; Shultz et al., in press).

We designed arithmetic comparison tasks that provided interesting similarities to some of the psychological simulations. In particular, instead of simply adding or multiplying, the nets learn to compare sums or products to some value and then output whether the sum or product is greater than, less than, or equal to that comparative value.

The addition and multiplication tasks each involve three linear input units. The first two input units each code a randomly selected integer in the range from 0 to 9, inclusive. The third input unit codes a randomly selected comparison integer. For addition problems, the comparison values are in the range of 0 to 19, inclusive; for multiplication the range is 0 to 82, inclusive. Two output units code the results of the comparison. Target outputs of 0.5 and -0.5 represent that the results of the arithmetic operation are *greater than* the comparison value, targets of -0.5 and 0.5 represent *less than*, and targets of 0.5 and 0.5 represent *equal to*. For problems involving both addition and multiplication, a fourth input unit codes the type of arithmetic operation to be performed: 0 for addition, 1 for multiplication.

Nets trained on either addition or multiplication have 100 randomly selected training patterns, with the restriction that 45 of them have correct answers of *greater than*, 45 have correct answers of *less than*, and 10 have correct answers of *equal to*. The latter constraints are designed to reduce the natural skew of comparative values in the high direction on multiplication problems. Nets trained on both addition and multiplication have 100 randomly selected addition problems and 100 randomly selected multiplication problems, subject to the constraints just described. We trained three nets on addition, three on multiplication, and three on both addition and multiplication.

## 6.1    RESULTS FOR ADDITION

PCA of contributions in all three addition nets yield two significant components. In each of the three nets, the component scores form three clusters, representing the three correct answers. In all three nets, the first component distinguishes *greater than* from *less than* answers and places *equal to* answers in the middle; the second component distinguishes

*equal to* from *unequal to* answers. The primary role of the hidden unit in these nets is to distinguish equality from inequality. The hidden unit is not required to perform addition per se in these nets, which have additive activation functions.

## 6.2    RESULTS FOR MULTIPLICATION

PCA applied to the contributions in the three multiplication nets yields from 3 to 4 significant components. Plots of rotated component scores show that the first component separates *greater than* from *less than* outputs, placing *equal to* outputs in the middle. Other components further differentiate the problems in these categories into several smaller groups that are related to the particular values being multiplied. Rotated component loadings indicate that component 1 is associated not only with contributions coming from the bias unit and the input units, but also with contributions from some hidden units. This underscores the need for hidden units to capture the non-linearities inherent to multiplication.

## 6.3    RESULTS FOR BOTH ADDITION AND MULTIPLICATION

PCA of contributions yields three components in each of the three nets taught to do both addition and multiplication. In addition to the familiar distinctions between *greater than*, *less than*, and *equal to* outputs found in nets doing either addition or multiplication, it is of interest to determine whether nets doing both operations distinguish between adding and multiplying.

Figure 1 shows the rotated component scores for net 1. Components 1 and 2 (accounting for 30.2% and 21.9% of the variance, respectively) together distinguish *greater than* answers from the rest. Component 3, accounting for 20.2% of the variance, separates *equal to* answers from *less than* answers and multiplication from addition for *greater than* answers. Together, components 2 and 3 separate multiplication from addition for *less than* answers. Results for the other two nets learning both multiplication and addition comparisons are essentially similar to those for net 1.

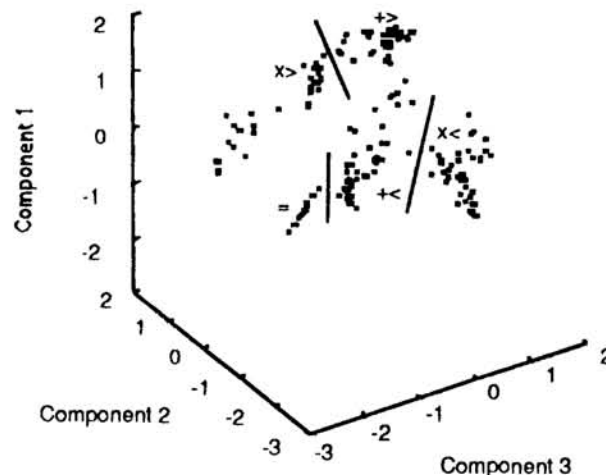

Figure 1. Rotated component scores for a net doing both addition and multiplication.

## 6.4    DISCUSSION OF COMPARATIVE ARITHMETIC

As with continuous XOR, there is considerable variation among networks learning comparative arithmetic problems. Varying numbers of hidden units are recruited by the networks and different types of components emerge from PCA of network contributions. In some cases, clear roles can be assigned to particular components, but in other cases, separation of input patterns relies on interactions among the various components.

Yet with all of this variation, it is apparent that the nets learn to separate arithmetic problems according to features afforded by the training set. Nets learning either addition or multiplication differentiate the problems according to answer types: *greater than*, *less than*, and *equal to*. Nets learning both arithmetic operations supplement these answer distinctions with the operational distinction between adding and multiplying.

## 7   APPLICATION TO THE TWO-SPIRALS PROBLEM

We next apply contribution analysis to a particularly difficult discrimination problem requiring a relatively large number of hidden units. The two-spirals problem requires the net to distinguish between two interlocking spirals that wrap around their origin three times. The standard version of this problem has two sets of 97 continuous-valued $x, y$ pairs, each set representing one of the spirals. The difficulty of the two-spirals problem is underscored by the finding that standard back-propagation nets are unable to learn it (Wieland, unpublished, cited in Fahlman & Lebiere, 1990). The best success to date on the two-spirals problem was reported with cascade-correlation nets, which learned in an average of 1700 epochs while recruiting from 12 to 19 hidden units (Fahlman & Lebiere, 1990). The relative difficulty of the two-spirals problem is undoubtedly due to its high degree of non-linearity. It suited our need for a relatively difficult, but fairly well understood problem on which to apply contribution analysis. We ran three nets using the 194 continuous $x, y$ pairs as inputs and a single sigmoid output unit, signaling -0.5 for spiral 1 and 0.5 for spiral 2.

Because of the relative difficulty of interpreting plots of component scores for this problem, we focus primarily on the extreme component scores, defined as less than -1 or greater than 1. Those $x, y$ input pairs with extreme component scores on the first two components for net 1 are plotted in Figure 2 as filled points on the two spirals. There are separate plots for the positive and negative ends of each of the two components. The filled points in each quadrant of Figure 2 define a shape resembling a tilted hourglass covering approximately one-half of the spirals. The positive end of component 1 can be seen to focus on the northeast sector of spiral 1 and the southwest sector of spiral 2. The negative end of component 1 has an opposite focus on the northeast sector of spiral 2 and the southwest sector of spiral 1. Component 2 does precisely the opposite of component 1: its positive end deals with the southeast sector of spiral 1 and the northwest sector of spiral 2 and its negative end deals with the southeast sector of spiral 2 and the northwest sector of spiral 1. Comparable plots for the other two nets show this same hourglass shape, but in a different orientation.

The networks appear to be exploiting the symmetries of the two spirals in reaching a solution. Examination of Figure 2 reveals the essential symmetries of the problem. For each $x, y$ pair, there exists a corresponding $-x, -y$ pair 180 degrees opposite and lying on the other spiral. Networks learn to treat these mirror image points similarly, as revealed by the fact that the plots of extreme component scores in Figures 2 are perfectly symmetrical across the two spirals. If a point on one spiral is plotted, then so is the corresponding point on the other spiral, 180 degrees opposite and at the same distance out from the center of the spirals. If a trained network learns that a given $x, y$ pair is on spiral 1, then it also seems to know that the $-x, -y$ pair is on spiral 2. Thus, it make good sense for the network to represent these opposing pairs similarly.

Recall that contributions are scaled by the sign of their targets, so that all of the products of sending activations and output weights for spiral 1 are multiplied by -1. This is to ensure that contributions bring output unit activations close to their targets in proportion

to the size of the contribution. Ignoring this scaling by target, the networks possess sufficient information to separate the two spirals even though they represent points of the two spirals in similar fashion. The plot of the extreme component scores in Figure 2 suggests that the critical information for separating the two spirals derives mainly from the signs of the input activations.

Because scaling contributions by the sign of the output target appears to obscure a full picture of network solutions to the two-spirals problem, there may be some value in using unscaled contributions in network analysis. Use of unscaled contributions also could be justified on the grounds that the net has no knowledge of targets as it represents a particular problem; target information is only used in the error correction process. A disadvantage of using unscaled contributions is that one cannot distinguish contributions that facilitate vs. contributions that inhibit reaching a relatively error free solution.

The symmetry of these network representations suggests a level of systematicity that is, on some accounts, not supposed to be possible in neural nets (Fodor & Pylyshyn, 1988). Whether this representational symmetry reflects systematicity in performance is another matter. One empirical prediction would be that as a net learns that $x$, $y$ is on one spiral, it also learns at about the same time that $-x$, $-y$ is on the other spiral. If confirmed, this would demonstrate a clear case of systematic cognition in neural nets.

# 8    GENERAL DISCUSSION

Performing PCA on network contributions is here shown to be a useful technique for understanding the performance of networks constructed by the cascade-correlation learning algorithm. Because cascade-correlation nets typically possess multiple hidden layers and are fully cross connected, they are difficult to analyze with more standard methods emphasizing activation patterns on the hidden units alone. Examination of their weight patterns is also problematic, particularly in larger networks, because of the highly distributed nature of the net's representations.

Analyzing contributions, in contrast to either hidden unit activations or weights, is a naturally appealing solution. Contributions capture the influence coming into output units both from adjacent hidden units and from distant, cross connected hidden and input units. Moreover, because contributions include both sending activations and connecting weights, they are not unduly sensitive to one at the expense of the other.

In the three domains examined in the present paper, PCA of the network contributions both confirm some expected results and provide new insights into network performance. In all cases examined, the nets succeed in drawing all of the important distinctions in their representations that are afforded by the training patterns, whether these distinctions concern the type of output or the operation being performed on the input. In combination with further experimentation and analysis of network weights and activation patterns, this technique could help to provide an account of how networks accomplish whatever it is they learn to accomplish.

It might be of interest to apply the present technique at various points in the learning process to obtain a developmental trace of network performance. Would all networks learning under the same constraints progress through the same stages of development, in terms of the problem distinctions they are able to make? This would be of particular interest to network simulations of human cognitive development, which has been claimed to be stage-like in its progressions.

The present technique could also be useful in predicting the results of lesioning experiments on neural nets. If the role of a hidden unit can be identified by its association with a particular principal component, then it could be predicted that lesioning this unit would impair the function served by the component.

## Acknowledgments

This research was supported by the Natural Sciences and Engineering Research Council of Canada and the MacArthur Foundation. Helpful comments were provided by Scott Fahlman, Denis Mareschal, Yuriko Oshima-Takane, and Sheldon Tetewsky.

## References

Cattell, R. B. (1966). The scree test for the number of factors. *Multivariate Behavioral Research*, 1, 245-276.

Elman, J. L. (1989). Representation and structure in connectionist models. CRL Technical Report 8903, Center for Research in Language, University of California at San Diego.

Elman, J. L. (1990). Finding structure in time. *Cognitive Science*, 14, 179-211.

Fahlman, S. E., & Lebiere, C. (1990.) The Cascade-Correlation learning architecture. In D. Touretzky (Ed.), *Advances in neural information processing systems 2*, (pp. 524-532). Mountain View, CA: Morgan Kaufmann.

Flury, B. (1988). *Common principal components and related multivariate models*. New York: Wesley.

Fodor, J., & Pylyshyn, Z. (1988). Connectionism and cognitive architecture: A critical analysis. *Cognition*, 28, 3-71.

Hinton, G. E., & Sejnowski, T. J. (1986). Learning and relearning in Boltzmann machines. In D. E. Rumelhart & J. L. McClelland (Eds.), *Parallel distributed processing: Explorations in the microstructure of cognition, Volume 1: Foundations*, pp. 282-317. Cambridge, MA: MIT Press.

Lang, K. J., & Witbrock, M. J. (1988). Learning to tell two spirals apart. In D. Touretzky, G. Hinton, & T. Sejnowski (Eds)., *Proceedings of the Connectionist Models Summer School*, (pp. 52-59). Mountain View, CA: Morgan Kaufmann.

McClelland, J. L. (1989). Parallel distributed processing: Implications for cognition and development. In Morris, R. G. M. (Ed.), *Parallel distributed processing: Implications for psychology and neurobiology*, pp. 8-45. Oxford University Press.

Rumelhart, D. E., Hinton, G. E., & Williams, R. J. (1986). Learning internal representations by error propagation. In D. E. Rumelhart & J. L. McClelland (Eds.), *Parallel distributed processing: Explorations in the microstructure of cognition, Volume 1: Foundations*, pp. 318-362. Cambridge, MA: MIT Press.

Sanger, D. (1989). Contribution analysis: A technique for assigning responsibilities to hidden units in connectionist networks. *Connection Science*, 1, 115-138.

Sejnowski, T. J., & Rosenberg, C. R. (1987). Parallel networks that learn to pronounce English text. *Complex Systems*, 1, 145-168.

Shultz, T. R., Schmidt, W. C., Buckingham, D., & Mareschal, D. (In press). Modeling cognitive development with a generative connectionist algorithm. In G. Halford & T. Simon (Eds.), *Developing cognitive competence: New approaches to process modeling*. Hillsdale, NJ: Erlbaum.

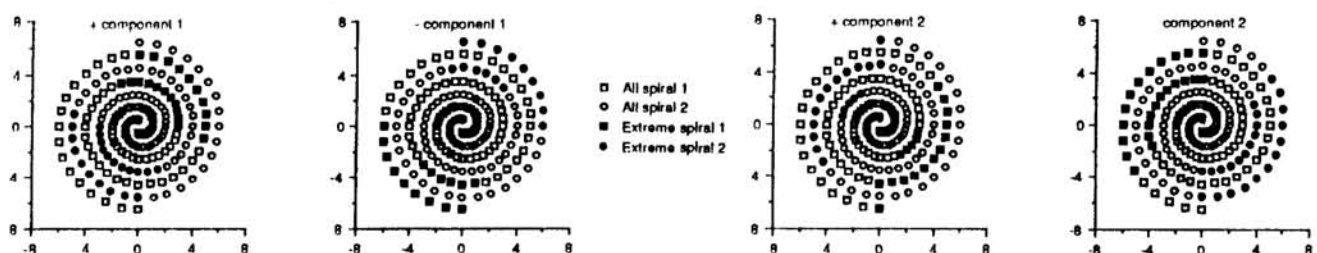

Figure 2. Extreme rotated component scores for a net on the two-spirals problem.